# Natural Dolphin Echo Recognition Using an Integrator Gateway Network

**Herbert L. Roitblat**
Department of Psychology, University
of Hawaii, Honolulu, HI 96822

**Patrick W. B Moore, Paul E. Nachtigall, & Ralph H. Penner**
Naval Ocean Systems Center, Hawaii
Laboratory, Kailua, Hawaii, 96734

## Abstract

We have been studying the performance of a bottlenosed dolphin on a delayed matching-to-sample task to gain insight into the processes and mechanisms that the animal uses during echolocation. The dolphin recognizes targets by emitting natural sonar signals and listening to the echoes that return. This paper describes a novel neural network architecture, called an integrator gateway network, that we have developed to account for this performance. The integrator gateway network combines information from multiple echoes to classify targets with about 90% accuracy. In contrast, a standard backpropagation network performed with only about 63% accuracy.

## 1. INTRODUCTION

The study of animals can provide a very important source of information for the design of automated artificial systems such as robots and autonomous vehicles. Animals have evolved in a real world, solving real problems, such as gathering and interpreting essential information. We call the process of using animal studies to inform the design of artificial systems biomimetics because the artificial systems are designed as mimics of biological ones.

## 2. INVESTIGATIONS OF DOLPHIN ECHOLOCATION PERFORMANCE

Dolphin echolocation clicks emerge from the rounded forehead or melon as a highly directional sound beam with 3 dB (half power) beamwidths of approximately 10° in both the vertical and horizontal planes (Au, et al., 1986). Echolocation clicks have peak energy at frequencies from 40 to 130 kHz with source levels of 220 dB re: $1 \mu$ Pa at 1 m (Au, 1980; Moore & Pawloski, 1990). Bottlenosed dolphins have excellent directionally selective hearing (Au & Moore, 1984), spanning over 7 octaves, and can detect frequencies as high as 150 kHz (Johnson, 1966).

## 3. BEHAVIORAL METHODS

We have been studying the performance of a bottlenosed dolphin on an echolocation delayed matching-to-sample (DMTS) task (e.g., Nachtigall, 1980; Nachtigall, et al., 1985; Roitblat, et al., 1990a; Moore, et al., 1990). In this task a sample stimulus is presented underwater to a blindfolded dolphin. The dolphin is allowed to echolocate on this object ad lib. The object is then removed from the water, and after a short delay, three alternative objects are presented (the comparison stimuli). One of these objects is identical to (matches) the sample object, and the dolphin is required to indicate the matching stimulus by touching a response wand in front of it. The object that serves as sample and the location of the correct match vary randomly from trial to trial.

Recent work has concentrated on performance with three sample and comparison stimuli: (a) a PVC plastic tube, (b) a water-filled stainless steel sphere, and (c) a solid aluminum cone (see Roitblat, et al., 1990a). On average the dolphin used 37.2 clicks to identify the sample, and an average of 4.2 scans to examine the three comparison stimuli. A scan is a train of clicks to a single stimulus ended either by the initiation of a scan to another stimulus or by a cessation of clicking

The dolphin's scanning patterns were modeled using sequential sampling theory (see also Roitblat, 1984). Simulations based on this model provide a reasonably good approximation of the dolphin's performance (Roitblat, et al., 1990a). The simulation differed from the dolphin's actual performance, however, in that it was less variable than the live dolphin. We return to the problem of accounting for this difference in variability below after considering some models of the details of echo recognition.

## 4. ARTIFICIAL NEURAL NETWORKS

We have developed a series of neural-network models of dolphin echolocation processing (see also Gorman and Sejnowski, 1988). We (Moore, et al., 1990; Roitblat, et al., 1989) trained a counterpropagation network (Hecht-Nielsen, 1987, 1988) to classify echoes represented by their spectra into categories corresponding to each of the stimuli in our current stimulus set. The network correctly classified more than 95% of these spectra. This classification suggests two things. First, the spectral information

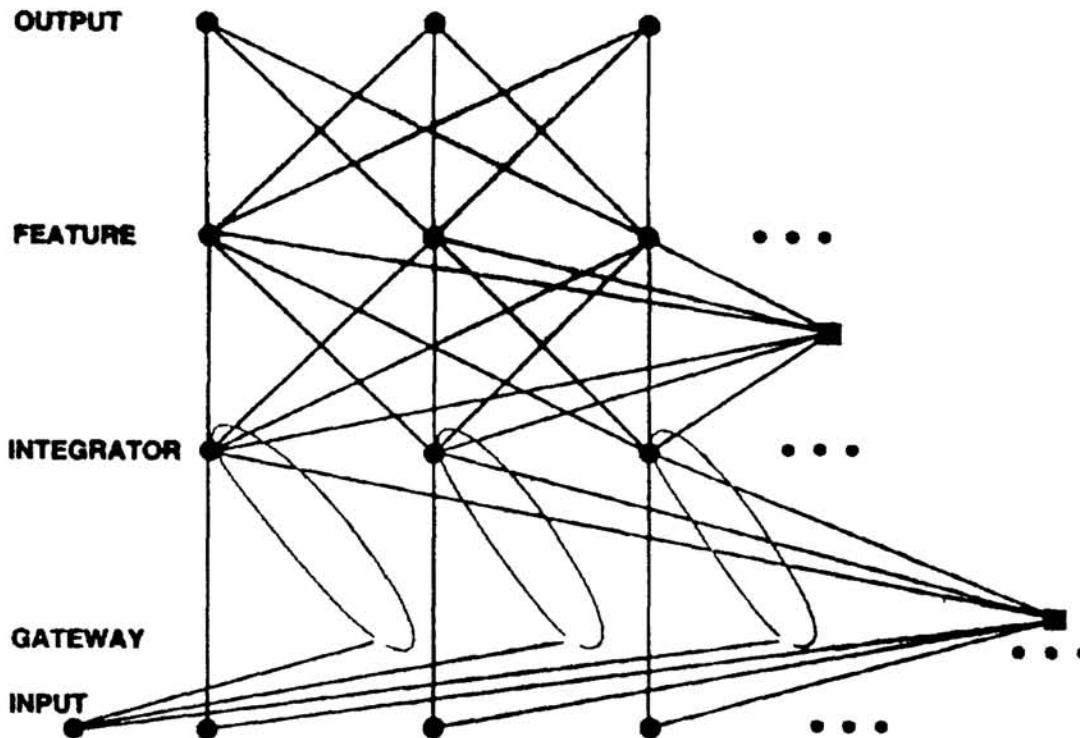

Figure    1.    A    schematic    of    the    Integrator    Gateway    Network.

present in the echoes was sufficient to identify the targets on which the dolphin was echolocating. Second, only a single echo was necessary to classify the target. Although the network could identify the target with only a single echo, the dolphin concurrently performing the same task emitted many clicks in identifying the same targets. Further investigation revealed that the clicks emitted by the dolphin were more variable than our initial sample suggested (Roitblat, et al., 1990b). This variability provides one possible explanation for the high performance level, and low variability of our initial model.

## 4.1 THE INTEGRATOR GATEWAY NETWORK

Our integrator gateway network incorporates features of the sequential sampling model described earlier, including the assumptions that the dolphin averages or sums spectral information from successive echoes and continues to emit clicks and collect returning echoes until it can classify the target producing those echoes with sufficient confidence. It mimics the dolphin's strategy of using multiple echoes to identify each target. Figure 1 shows schematic of the Integrator Gateway Network.

Network inputs were 30-dimensional spectral vectors containing echo amplitudes in 1.95 kHz wide frequency bins. The echoes were captured and digitized during the dolphin's matching-to-sample performance. In addition to the 30 bins of spectral information, each echo was also marked as to whether the echo was (1.00) or was not

(0.00) at the start of an echo train. Recall that the dolphin directs a series of clicks to one target at a time, so it seemed plausible to include information marking the start of a click train. The frequency inputs were then passed to a scalar unit and to the integrator layer. The integrator layer also contained 30 units, connected to the frequency units in the input layer in a corresponding one-to-one pattern. The connections to the scalar unit were fixed at $1/n$, where n is the number of frequency inputs. The weights to the integrator layer were fixed at 1.00. The output of the scalar unit, i.e., the sum of all of its inputs, was passed to each unit in the integrator layer via a fixed weight of -1.00. The effect of this scalar unit was to subtract the average activity of the input layer (neglecting the start-of-train marker) from the inputs to the integrator layer. This subtraction preserved all of the relative activity information present in the inputs, but kept the inputs within a manageable range.

The elements in the integrator layer computed a cumulative sum of the inputs they received. The role of this layer was to accumulate and integrate information from successive echo spectra. The outputs of the integrator layer were passed via fixed connections with 1.00 weights to corresponding units in the gateway layer. The integrator layer and the gateway layer each contained the same number of units. Each unit in the gateway layer acted as a reset for the corresponding unit in the integrator layer, and connected back to its corresponding unit with a weight of -1.00. Each unit in the gateway layer employed a multiplicative transfer function that multiplied the input from its corresponding unit in the integrator layer with the value of the start-of-train marker. Because this marker had 1.00 activity at the start of a scan and 0.00 activity otherwise, it functioned as a reset signal, causing the units in the integrator layer to be reset to 0.00 at the start of every scan; their previous activation level was subtracted from their input.

The output of the integrator layer also led via variable-weight connections to each of the elements in the feature layer. The same kind of scalar unit that intervened between the input layer and integrator layer was also used between the integrator layer and feature layer to subtract the average activity of the integrator layer, again to keep activations within a manageable range. The outputs of the feature layer led via variable-weight connections to the classifier layer. The elements in these two layers contained sigmoid transfer functions and were trained using a standard cumulative back-propagation algorithm with the epoch duration set to the number of training samples (60).

The training set consisted of six sets of ten successive echoes each, selected from the ends of haphazardly chosen echo trains. An equal number of cone, tube, and sphere echoes were used. The training set was a relatively small subset (4%) of the total set of available echoes (1,335).

## 4.2 INTEGRATOR GATEWAY RESULTS AND DISCUSSION

Figure 2 shows the results of generalization testing of the network in the form of a derived confidence measure. The network was given all 30 scans (10 scans of each tar-

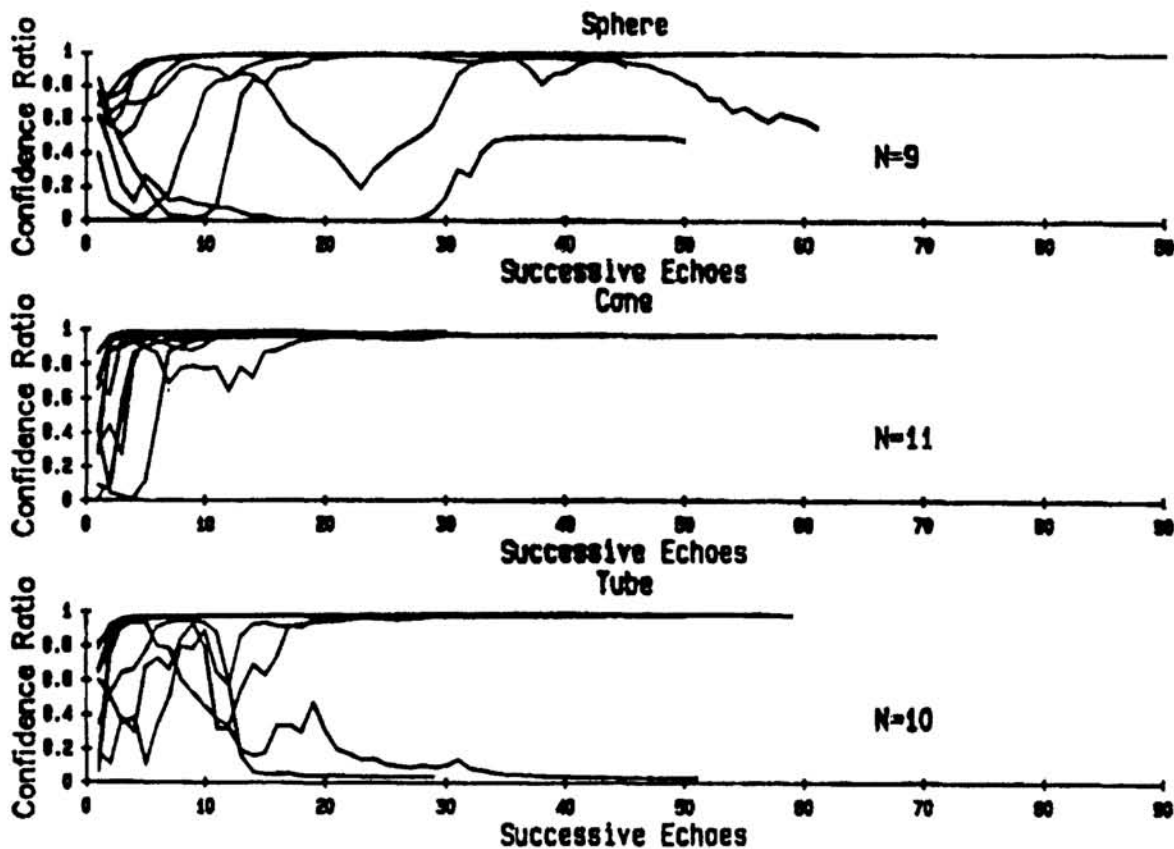

Figure 2. Results of generalization testing of the network in the form of the confidence of the network in assigning the echo train to the proper category. See text.

get for a total of 1,335 sequential echoes), and was required to classify each echo train. "Confidence" was defined as the ratio of the activation level of the correct classification versus the total output of the three classification units. A confidence ratio of 1.00 indicates that only the correct unit is active. Confidence of 0.00 indicates that the correct unit is entirely inactive. Intermediate confidences correspond to intermediate likelihood ratios (Qian & Sejnowski, 1988).

Recall that echo trains varied in length under control of the dolphin. Therefore, it is not entirely clear how to measure the network performance. According to sequential sampling theory (see Roitblat, et al., 1990a) a rational decision maker collects echo evidence only until a sufficiently confident classification is available and then stops. Table 1 shows the number of clicks in each train that were required to reach a confidence ratio of 0.96 and the classification that the network derived. Some of the scans ended before the network could achieve this confidence level. Three erroneous classifications were made (90% correct).

Table 1
Number of Clicks to Network Confidence Criterion

| Target Scanned | | | | | |
|---|---|---|---|---|---|
| Sphere | Cone | Tube | Sphere | Cone | Tube |
| Integrator Gateway | | | Backpropagation | | |
| 16S | 20C | 40C | 1S | 1C | 3S |
| 9S | 4C | 18C | 6S | 30C | 1S |
| 7S | 2C | 20T | 1S | 1C | 11S[1] |
| 6S | 6C | 23T | 5S | 2C | 1T |
| 19S | 14C | 5T | 14S | 2C | 14T |
| 19S | 6C | 4T | 14S | 30S[1] | 1T |
| 34S | 6C | 4T | 3S | 32S[1] | 1T |
| 7S | 4C | 4T | 1S | 57S | 1T |
| 23C | 6C | 5T | 40T | 22S | 2T |
| | 3C | 4T | | 22S | 1T |
| | 11C | | | 27T | |

*Note*: Entries are the number of clicks needed by the network to achieve the 0.96 confidence criterion. C indicates a Cone decision, S indicates a Sphere decision, T indicates a Tube decision. [1]Indicates that the dolphin stopped echolocating before the network reached its confidence criterion. On these scans, the decision is the one with the highest confidence at the end of the scan.

## 4.3 A SIMPLE BACKPROPAGATION NETWORK

The integrator gateway network reflects the assumption of sequential sampling theory that the dolphin combines information from successive echoes in deriving its identification. In contrast, a standard backpropagation network does not integrate over successive echoes, but instead attempts to identify each echo independently. A backpropagation network can be used as a model of a system that emits multiple clicks because the echoes vary in quality. Rather than integrating the echoes, it simply waits for a single adequate echo that allows it to meet its confidence criterion.

We trained a backpropagation network (using the fast-backpropagation algorithm to adjust the weights (Samad, 1988) on the same data that were submitted to the integrator network in order to determine whether the additional structure of the integrator network contributed to its performance accuracy. The network contained exactly the same number of inputs, hidden units, outputs, and adjustable connections as the integrator network. The networks differed only in absence of the integration apparatus in the backpropagation network.

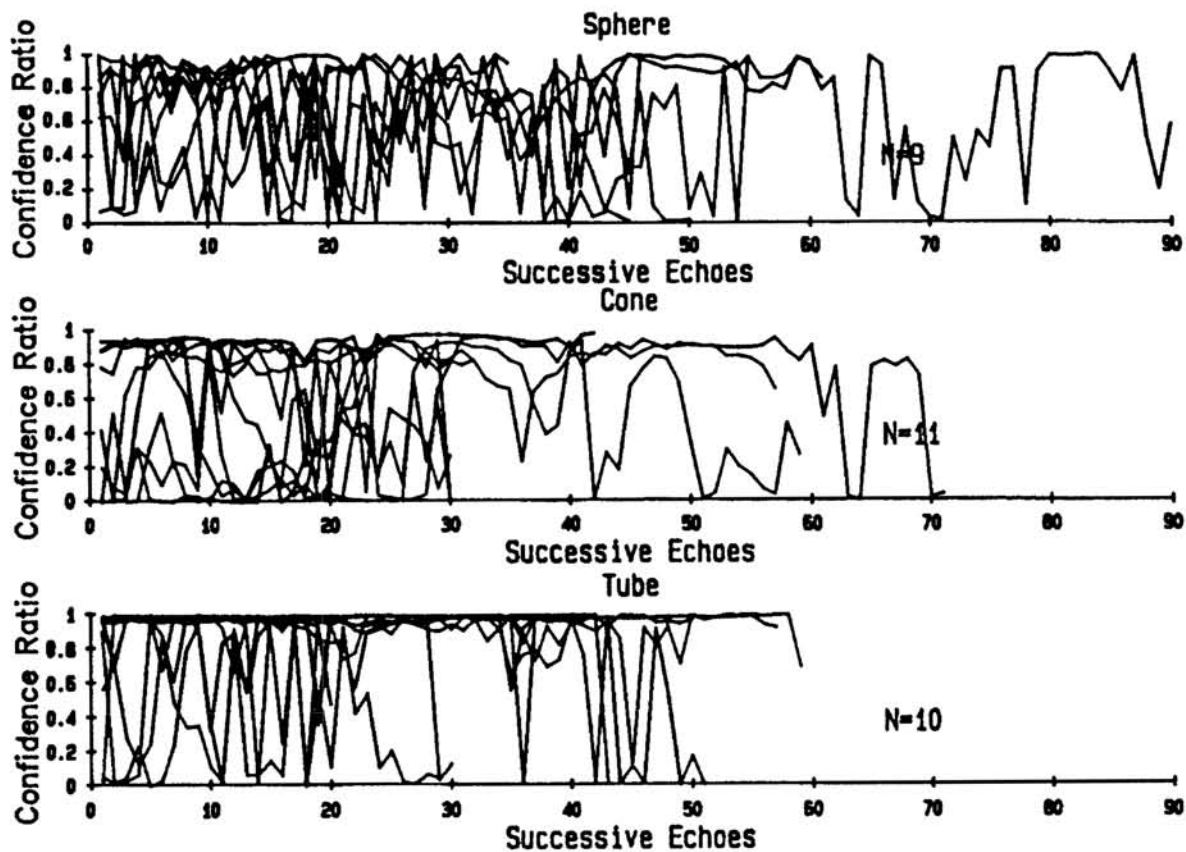

Figure 3. Confidence of the backpropagation network in assigning the echo train to the proper category as a function of the number of echoes received.

## 4.4 BACKPROPAGATION RESULTS

Figure 3 shows the confidence of the backpropagation network in assigning the echo train to the proper category as a function of the number of echoes received. Compared to the categorization performance of the integrator network, the backpropagation network was much more variable. As Figure 3 shows, the individual echoes were highly variable, and frequently assigned to an erroneous category.

The performance of the backpropagation network when judged by the standards of sequential sampling theory are also shown in Table 1. This table shows the number of clicks necessary to first reach a classification with greater then 0.96 confidence. On average the backpropagation network (11.57 echoes) reached its confidence criterion in the about the same number of clicks (t (df = 58) = 0.03, p> .05) as the integrator network (11.67 echoes), but it produced more errors ($X^2$ (df=1) = 5.96).

These data suggest that the integrator network added significantly to the ability to classify sequentially produced echoes. By implementing a signal "averaging" mechanism in the neural network the system could take advantage of the redundancy inherent in the use of multiple echoes from the same source and in the stochastic properties of the noise in which those echoes are embedded. In contrast, the backpropaga-

tion network is required to process not only the characteristics of the echoes themselves, but also the characteristics of the noise. This results in many spurious classifications.

The gateway integrator network adds a level of complexity to the standard backpropagation network architecture that contributes substantially to its performance. Its design is inspired by properties of the dolphin's performance (Nachtigall & Moore, 1988) and it represents one step along a development path that seeks to include more and more of the mechanisms that we can identify from the neurobiology of echolocation and from the performance of dolphins in their aquatic environment.

## References

Au, W. W. L. (1980). Echolocation signals of the Atlantic bottlenose dolphin (Tursiops truncatus) in open waters. In R. G. Busnel & J. F. Fish (Eds.) *Animal sonar systems*. (pp. 251-282). New York: Plenum Press.

Au, W. W. L. & Moore, P. W. B. (1984). Receiving beam patterns and directivity indices of the Atlantic bottlenose dolphin *Tursiops truncatus*. *Journal of the Acoustical Society of America*, 75, 255-262.

Au, W. W. L., Moore, P. W. B. & Pawloski, D. (1986). Echolocating transmitting beam of the Atlantic bottlenose dolphin. *Journal of the Acoustical Society of America*, 80, 688-691.

Gorman, R. P. & Sejnowski, T. J. (1988). Analysis of hidden units in a layered network trained to classify sonar targets. *Neural Networks*, 1, 75-89.

Hecht-Nielsen, R. (1987). Counterpropagation networks. *Applied Optics*, 26, 4979-4984.

Hecht-Nielsen, R. (1988). Applications of counterpropagation networks. *Neural Networks*, 1, 131-139.

Johnson, C. S. (1966). *Auditory thresholds of the bottlenosed porpoise, Tursiops truncatus (Montague)* (Naval Ordnance Test Station Technical Publication No 4178). Naval Ordnance Test Station.

Moore, P. W. B. & Pawloski, D. A. (1990). Investigations on the control of echolocation pulses in the dolphin (*Tursiops truncatus*). In J. Thomas & R. Kastelein (Eds.) *Sensory abilities of cetaceans*. New York: Plenum. In press.

Moore, P. W. B., Roitblat, H. L., Penner, R. H., & Nachtigall, P. E., Recognizing Successive Dolphin Echoes with an Integrator Gateway Network. Submitted for publication.

Nachtigall, P. E. (1980). Odontocete echolocation performance on object size, shape, and material, In R. G. Busnel & J. F. Fish (Eds.), *Animal Sonar Systems*, pp. 71-95, New York, Plenum Press.

Nachtigall, P. E., & Moore, P. W. B. (Eds.) (1988). *Animal sonar: Processes and performance*. New York: Plenum.

Nachtigall, P. E., Patterson, S. A., & Bauer, G. B. (1985). Echolocation delayed matching-to-sample in a bottlenose dolphin. Paper presented at the Sixth Biennial Conference on the Biology of Marine Mammals, Vancouver, B.C., Canada. November.

Qian, N. & Sejnowski, T. J. (1988). Predicting the secondary structure of globular proteins using neural network models. *Journal of Molecular Biology*, 202, 865-884.

Roitblat, H. L. (1984). Representations in pigeon working memory, In: H. L. Roitblat, T. G. Bever and H. S. Terrace (Eds.), *Animal cognition*. Hillsdale, NJ: Erlbaum, 79-97.

Roitblat, H. L., Moore, P. W. B., Nachtigall, P. E., Penner, R. H., & Au, W. W. L. (1989). Dolphin echolocation: Identification of returning echoes using a counterpropagation network. *Proceedings of the First International Joint Conference on Neural Networks*. Washington, DC: IEEE Press.

Roitblat, H. L., Penner, R. H. & Nachtigall, P. E. (1990a). Matching-to-sample by an echolocating dolphin. *Journal of Experimental Psychology: Animal Behavior Processes*, 16, 85-95.

Roitblat, H. L., Penner, R. H. & Nachtigall, P. E. (1990b). Attention and decision making in echolocation matching-to-sample by a bottlenose dolphin (*Tursiops truncatus*): the microstructure of decision making. In J. Thomas & R. Kastelein (Eds.) *Sensory abilities of cetaceans*. New York: Plenum. In press.

Samad, T. (1988). Back propagation is significantly faster if the expected value of the source unit is used for update. *International Neural Network Society Conference Abstracts*.
